# Learning to Learn with Compound HD Models

**Ruslan Salakhutdinov**
Department of Statistics, University of Toronto
rsalakhu@utstat.toronto.edu

**Joshua B. Tenenbaum**
Brain and Cognitive Sciences, MIT
jbt@mit.edu

**Antonio Torralba**
CSAIL, MIT
torralba@mit.edu

## Abstract

We introduce HD (or "Hierarchical-Deep") models, a new compositional learning architecture that integrates deep learning models with structured hierarchical Bayesian models. Specifically we show how we can learn a hierarchical Dirichlet process (HDP) prior over the activities of the top-level features in a Deep Boltzmann Machine (DBM). This compound HDP-DBM model learns to learn novel concepts from very few training examples, by learning low-level generic features, high-level features that capture correlations among low-level features, and a category hierarchy for sharing priors over the high-level features that are typical of different kinds of concepts. We present efficient learning and inference algorithms for the HDP-DBM model and show that it is able to learn new concepts from very few examples on CIFAR-100 object recognition, handwritten character recognition, and human motion capture datasets.

## 1 Introduction

"Learning to learn", or the ability to learn abstract representations that support transfer to novel but related tasks, lies at the core of many problems in computer vision, natural language processing, cognitive science, and machine learning. In typical applications of machine classification algorithms today, learning curves are measured in tens, hundreds or thousands of training examples. For humans learners, however, just one or a few examples are often sufficient to grasp a new category and make meaningful generalizations to novel instances [25, 16]. The architecture we describe here takes a step towards this "one-shot learning" ability by learning several forms of abstract knowledge that support transfer of useful representations from previously learned concepts to novel ones.

We call our architectures *compound HD models*, where "HD" stands for "Hierarchical-Deep", because they are derived by composing hierarchical nonparametric Bayesian models with deep networks, two influential approaches from the recent unsupervised learning literature with complementary strengths. Recently introduced deep learning models, including Deep Belief Networks [5], Deep Boltzmann Machines [14], deep autoencoders [10], and others [12, 11], have been shown to learn useful distributed feature representations for many high-dimensional datasets. The ability to automatically learn in multiple layers allows deep models to construct sophisticated domain-specific features without the need to rely on precise human-crafted input representations, increasingly important with the proliferation of data sets and application domains.

While the features learned by deep models can enable more rapid and accurate classification learning, deep networks themselves are not well suited to one-shot learning of novel classes. All units and parameters at all levels of the network are engaged in representing any given input and are adjusted together during learning. In contrast, we argue that one-shot learning of new classes will be easier in architectures that can explicitly identify only a small number of degrees of freedom (latent variables and parameters) that are relevant to the new concept being learned, and thereby achieve more appropriate and flexible transfer of learned representations to new tasks. This ability is the

hallmark of hierarchical Bayesian (HB) models, recently proposed in computer vision, statistics, and cognitive science [7, 25, 4, 13] for learning to learn from few examples. Unlike deep networks, these HB models explicitly represent category hierarchies that admit sharing the appropriate abstract knowledge about the new class's parameters via a prior abstracted from related classes. HB approaches, however, have complementary weaknesses relative to deep networks. They typically rely on domain-specific hand-crafted features [4, 1] (e.g. GIST, SIFT features in computer vision, MFCC features in speech perception domains). Committing to the a-priori defined feature representations, instead of learning them from data, can be detrimental. Moreover, many HB approaches often assume a fixed hierarchy for sharing parameters [17, 3] instead of learning the hierarchy in an unsupervised fashion.

In this work we investigate compound HD (hierarchical-deep) architectures that integrate these deep models with structured hierarchical Bayesian models. In particular, we show how we can learn a hierarchical Dirichlet process (HDP) prior over the activities of the top-level features in a Deep Boltzmann Machine (DBM), coming to represent both a layered hierarchy of increasingly abstract features, and a tree-structured hierarchy of classes. Our model depends minimally on domain-specific representations and achieves state-of-the-art one-shot learning performance by unsupervised discovery of three components: (a) low-level features that abstract from the raw high-dimensional sensory input (e.g. pixels, or 3D joint angles); (b) high-level part-like features that express the distinctive perceptual structure of a specific class, in terms of class-specific correlations over low-level features; and (c) a hierarchy of super-classes for sharing abstract knowledge among related classes. We evaluate the compound HDP-DBM model on three different perceptual domains. We also illustrate the advantages of having a full generative model, extending from highly abstract concepts all the way down to sensory inputs: we can not only generalize class labels but also synthesize new examples in novel classes that look reasonably natural, and we can significantly improve classification performance by learning parameters at *all levels jointly* by maximizing a joint log-probability score.

## 2 Deep Boltzmann Machines (DBMs)

A Deep Boltzmann Machine is a network of symmetrically coupled stochastic binary units. It contains a set of visible units $\mathbf{v} \in \{0,1\}^D$, and a sequence of layers of hidden units $\mathbf{h}^1 \in \{0,1\}^{F_1}$, $\mathbf{h}^2 \in \{0,1\}^{F_2}$,..., $\mathbf{h}^L \in \{0,1\}^{F_L}$. There are connections only between hidden units in adjacent layers, as well as between visible and hidden units in the first hidden layer. Consider a DBM with three hidden layers[1] (i.e. $L = 3$). The probability of a visible input $\mathbf{v}$ is:

$$P(\mathbf{v}; \psi) = \frac{1}{\mathcal{Z}(\psi)} \sum_{\mathbf{h}} \exp \left( \sum_{ij} \mathbf{W}_{ij}^{(1)} v_i h_j^1 + \sum_{jl} \mathbf{W}_{jl}^{(2)} h_j^1 h_l^2 + \sum_{lm} \mathbf{W}_{lm}^{(2)} h_l^2 h_m^3 \right), \qquad (1)$$

where $\mathbf{h} = \{\mathbf{h}^1, \mathbf{h}^2, \mathbf{h}^3\}$ are the set of hidden units, and $\psi = \{\mathbf{W}^{(1)}, \mathbf{W}^{(2)}, \mathbf{W}^{(3)}\}$ are the model parameters, representing visible-to-hidden and hidden-to-hidden symmetric interaction terms.

**Approximate Learning:** Exact maximum likelihood learning in this model is intractable, but efficient approximate learning of DBMs can be carried out by using a mean-field inference to estimate data-dependent expectations, and an MCMC based stochastic approximation procedure to approximate the model's expected sufficient statistics [14]. In particular, consider approximating the true posterior $P(\mathbf{h}|\mathbf{v}; \psi)$ with a fully factorized approximating distribution over the three sets of hidden units: $Q(\mathbf{h}|\mathbf{v}; \mu) = \prod_{j=1}^{F_1} \prod_{k=1}^{F_2} \prod_{m=1}^{F_3} q(h_j^1|\mathbf{v}) q(h_k^2|\mathbf{v}) q(h_m^3|\mathbf{v})$ where $\mu = \{\mu^1, \mu^2, \mu^3\}$ are the mean-field parameters with $q(h_i^l = 1) = \mu_i^l$ for $l = 1, 2, 3$. In this case, we can write down the variational lower bound on the log-probability of the data, which takes a particularly simple form:

$$\log P(\mathbf{v}; \psi) \quad \geq \quad \mathbf{v}^\top \mathbf{W}^{(1)} \mu^1 + {\mu^1}^\top \mathbf{W}^{(2)} \mu^2 + {\mu^2}^\top \mathbf{W}^{(3)} \mu^2 - \log \mathcal{Z}(\psi) + \mathcal{H}(Q), \quad (2)$$

where $\mathcal{H}(\cdot)$ is the entropy functional. Learning proceeds by finding the value of $\mu$ that maximizes this lower bound for the current value of model parameters $\psi$, which results in a set of the mean-field fixed-point equations. Given the variational parameters $\mu$, the model parameters $\psi$ are then updated to maximize the variational bound using stochastic approximation (for details see [14, 22, 26]).

**Multinomial DBMs:** To allow DBMs to express more information and introduce more structured hierarchical priors, we will use a conditional multinomial distribution to model activities of the top-level units. Specifically, we will use $M$ softmax units, each with "1-of-K" encoding (so that each

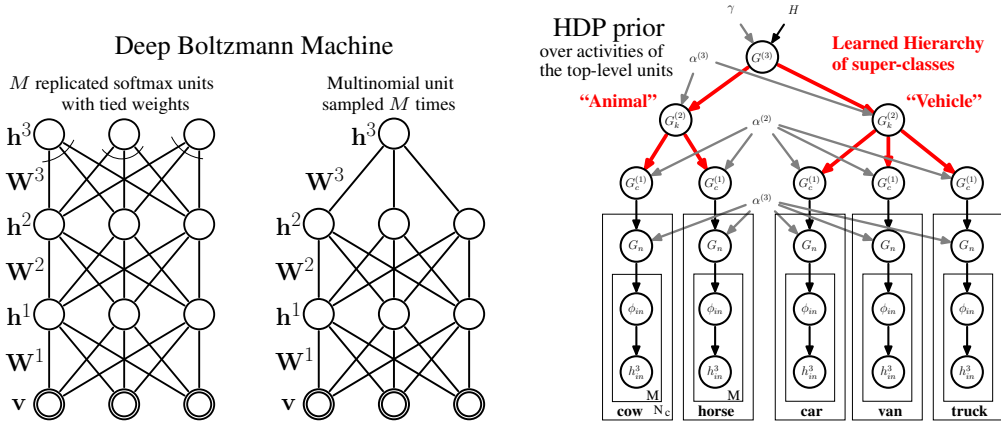

Figure 1: **Left:** Multinomial DBM model: the top layer represents M softmax hidden units $\mathbf{h}^3$, which share the same set of weights. **Middle:** A different interpretation: M softmax units are replaced by a single multinomial unit which is sampled M times. **Right:** Hierarchical Dirichlet Process prior over the states of $\mathbf{h}^3$.

unit contains a set of $K$ weights). All $M$ separate softmax units will share the same set of weights, connecting them to binary hidden units at the lower-level (Fig. 1). A key observation is that M separate copies of softmax units that all share the same set of weights can be viewed as a single multinomial unit that is samples M times [15, 19]. A pleasing property of using softmax units is that the mathematics underlying the learning algorithm for binary-binary DBMs remains the same.

## 3   Compound HDP-DBM model

After a DBM model has been learned, we have an undirected model that defines the joint distribution $P(\mathbf{v}, \mathbf{h}^1, \mathbf{h}^2, \mathbf{h}^3)$. One way to express what has been learned is the conditional model $P(\mathbf{v}, \mathbf{h}^1, \mathbf{h}^2|\mathbf{h}^3)$ and a prior term $P(\mathbf{h}^3)$. We can therefore rewrite the variational bound as:

$$\log P(\mathbf{v}) \geq \sum_{\mathbf{h}^1,\mathbf{h}^2,\mathbf{h}^3} Q(\mathbf{h}|\mathbf{v};\mu) \log P(\mathbf{v}, \mathbf{h}^1, \mathbf{h}^2|\mathbf{h}^3) + \mathcal{H}(Q) + \sum_{\mathbf{h}^3} Q(\mathbf{h}^3|\mathbf{v};\mu) \log P(\mathbf{h}^3). \quad (3)$$

This particular decomposition lies at the core of the greedy recursive pretraining algorithm: we keep the learned conditional model $P(\mathbf{v}, \mathbf{h}^1, \mathbf{h}^2|\mathbf{h}^3)$, but maximize the variational lower-bound of Eq. 3 with respect to the last term [5]. Instead of adding an additional undirected layer, (e.g. a restricted Boltzmann machine), to model $P(\mathbf{h}^3)$, we can place a hierarchical Dirichlet process prior over $\mathbf{h}^3$, that will allow us to learn category hierarchies, and more importantly, useful representations of classes that contain few training examples. The part we keep, $P(\mathbf{v}, \mathbf{h}^1, \mathbf{h}^2|\mathbf{h}^3)$, represents a *conditional* DBM model, which can be viewed as a two-layer DBM but with bias terms given by the states of $\mathbf{h}^3$:

$$P(\mathbf{v}, \mathbf{h}^1, \mathbf{h}^2|\mathbf{h}^3) = \frac{1}{\mathcal{Z}(\psi, \mathbf{h}^3)} \exp\left( \sum_{ij} \mathbf{W}_{ij}^{(1)} v_i h_j^1 + \sum_{jl} \mathbf{W}_{jl}^{(2)} h_j^1 h_l^2 + \sum_{lm} \mathbf{W}_{lm}^{(3)} h_l^2 h_m^3 \right). \quad (4)$$

### 3.1   A Hierarchical Bayesian Topic Prior

In a typical hierarchical topic model, we observe a set of $N$ documents, each of which is modeled as a mixture over topics, that are shared among documents. Let there be $K$ words in the vocabulary. A topic $t$ is a discrete distribution over $K$ words with probability vector $\phi_t$. Each document $n$ has its own distribution over topics given by probabilities $\theta_n$.

In our compound HDP-DBM model, we will use a hierarchical topic model as a prior over the activities of the DBM's top-level features. Specifically, the term "document" will refer to the top-level multinomial unit $\mathbf{h}^3$, and $M$ "words" in the document will represent the $M$ samples, or active DBM's top-level features, generated by this multinomial unit. Words in each document are drawn by choosing a topic $t$ with probability $\theta_{nt}$, and then choosing a word $w$ with probability $\phi_{tw}$. We will often refer to topics as our learned *higher-level features*, each of which defines a topic specific distribution over DBM's $\mathbf{h}^3$ features. Let $h_{in}^3$ be the $i^{th}$ word in document $n$, and $x_{in}$ be its topic:

$$\theta_n|\pi \sim \text{Dir}(\alpha\pi), \quad \phi_t|\tau \sim \text{Dir}(\beta\tau), \quad x_{in}|\theta_n \sim \text{Mult}(\theta_n), \quad h_{in}^3|x_{in}, \phi_{x_{in}} \sim \text{Mult}(\phi_{x_{in}}), \quad (5)$$

where $\pi$ is the global distribution over topics, $\tau$ is the global distribution over $K$ words, and $\alpha$ and $\beta$ are concentration parameters.

Let us further assume that we are presented with a fixed two-level category hierarchy. Suppose that $N$ documents, or objects, are partitioned into $C$ basic level categories (e.g. cow, sheep, car). We represent such partition by a vector $\mathbf{z}^b$ of length $N$, each entry of which is $z_n^b \in \{1, ..., C\}$. We also assume that our $C$ basic-level categories are partitioned into $S$ super-categories (e.g. animal, vehicle), represented by a vector $\mathbf{z}^s$ of length $C$, with $z_c^s \in \{1, ..., S\}$. These partitions define a *fixed two-level* tree hierarchy (see Fig. 1). We will relax this assumption later.

The hierarchical topic model can be readily extended to modeling the above hierarchy. For each document $n$ that belong to the basic category $c$, we place a common Dirichlet prior over $\theta_n$ with parameters $\pi_c^{(1)}$. The Dirichlet parameters $\pi^{(1)}$ are themselves drawn from a Dirichlet prior with parameters $\pi^{(2)}$, and so on (see Fig. 1). Specifically, we define the following prior over $\mathbf{h}^3$:

$$
\begin{align}
\pi_s^{(2)}|\pi_g^{(3)} &\sim \text{Dir}(\alpha^{(3)}\pi_g^3), \quad \text{for each super-category s=1,..,S} \tag{6}\\
\pi_c^{(1)}|\pi_{z_c^s}^{(2)} &\sim \text{Dir}(\alpha^{(2)}\pi_{z_c^s}^{(2)}), \quad \text{for each basic-category } c = 1, .., C\\
\theta_n|\pi_{z_n^b}^{(1)} &\sim \text{Dir}(\alpha^{(1)}\pi_{z_n^b}^{(1)}), \quad \text{for each document } n = 1, .., N\\
x_{in}|\theta_n &\sim \text{Mult}(\theta_n), \quad \text{for each word } i = 1, .., M\\
\phi_t|\tau &\sim \text{Dir}(\beta\tau), \quad h_{in}^3|x_{in}, \phi_{x_{in}} \sim \text{Mult}(\phi_{x_{in}}),
\end{align}
$$

where $\pi_g^{(3)}$ is the global distribution over topics, $\pi_s^{(2)}$ is the super-category specific and $\pi_c^{(1)}$ is the class specific distribution over topics, or higher-level features. These high-level features, in turn, define topic-specific distribution over $\mathbf{h}^3$ features, or "words" in a DBM model.

For a fixed number of topics $T$, the above model represents a hierarchical extension of LDA. We typically do not know the number of topics a-priori. It is therefore natural to consider a nonparametric extension based on the HDP model [21], which allows for a countably infinite number of topics. In the standard hierarchical Dirichlet process notation, we have

$$
G_g^{(3)} \sim \text{DP}(\gamma, \text{Dir}(\beta\tau)), \quad G_s^{(2)} \sim \text{DP}(\alpha^{(3)}, G_g^{(3)}), \quad G_c^{(1)} \sim \text{DP}(\alpha^{(2)}, G_{z_c^s}^{(2)}), \tag{7}
$$
$$
G_n \sim \text{DP}(\alpha^{(1)}, G_{z_n^b}^{(1)}), \quad \phi_{in}^*|G_n \sim G_n, \quad h_{in}^3|\phi_{in}^* \sim \text{Mult}(\phi_{in}^*),
$$

where $\text{Dir}(\beta\tau)$ is the base-distribution, and each $\phi^*$ is a factor associated with a single observation $h_{in}^3$. Making use of topic index variables $x_{in}$, we denote $\phi_{in}^* = \phi_{x_{in}}$ (see Eq. 6). Using a stick-breaking representation we can write: $G_g^{(3)}(\phi) = \sum_{t=1}^\infty \pi_{gt}^{(3)}\delta_{\phi_t}$, $G_s^{(2)}(\phi) = \sum_{t=1}^\infty \pi_{st}^{(2)}\delta_{\phi_t}$, $G_c^{(3)}(\phi) = \sum_{t=1}^\infty \pi_{ct}^{(1)}\delta_{\phi_t}$, and $G_n(\phi) = \sum_{t=1}^\infty \theta_{nt}\delta_{\phi_t}$ that represent sums of point masses. We also place Gamma priors over concentration parameters as in [21].

The overall generative model is shown in Fig. 1. To generate a sample we first draw $M$ words, or activations of the top-level features, from the HDP prior over $\mathbf{h}^3$ given by Eq. 7. Conditioned on $\mathbf{h}^3$, we sample the states of $\mathbf{v}$ from the conditional DBM model given by Eq. 4.

### 3.2 Modeling the number of super-categories

So far we have assumed that our model is presented with a two-level partition $\mathbf{z} = \{\mathbf{z}^s, \mathbf{z}^b\}$. If, however, we are not given any level-1 or level-2 category labels, we need to infer the distribution over the possible category structures. We place a nonparametric two-level nested Chinese Restaurant Prior (CRP) [2] over $\mathbf{z}$, which defines a prior over tree structures and is flexible enough to learn arbitrary hierarchies. The main building block of the nested CRP is the Chinese restaurant process, a distribution on partition of integers. Imagine a process by which customers enter a restaurant with an unbounded number of tables, where the $n^{th}$ customer occupies a table $k$ drawn from:

$$
P(z_n = k|z_1, ..., z_{n-1}) = \{\frac{n^k}{n-1+\eta}, \text{ if } n^k > 0; \quad \frac{\eta}{n-1+\eta}, \text{ if } k \text{ is new}\}, \tag{8}
$$

where $n^k$ is the number of previous customers at table $k$ and $\eta$ is the concentration parameter. The nested CRP, nCRP($\eta$), extends CRP to nested sequence of partitions, one for each level of the tree. In this case each observation $n$ is first assigned to the super-category $z_n^s$ using Eq. 8. Its assignment to the basic-level category $z_n^b$, that is placed under a super-category $z_n^s$, is again recursively drawn from Eq. 8. We also place a Gamma prior $\Gamma(1, 1)$ over $\eta$. The proposed model allows for both: a nonparametric prior over potentially unbounded number of global topics, or higher-level features, as well as a nonparametric prior that allow learning an arbitrary tree taxonomy.

# 4 Inference

Inferences about model parameters at all levels of hierarchy can be performed by MCMC. When the tree structure $\mathbf{z}$ of the model is not given, the inference process will alternate between fixing $\mathbf{z}$ while sampling the space of model parameters, and vice versa.

**Sampling HDP parameters:** Given category assignment vectors $\mathbf{z}$, and the states of the top-level DBM features $\mathbf{h}^3$, we use posterior representation sampler of [20]. In particular, the HDP sampler maintains the stick-breaking weights $\{\theta\}_{n=1}^N$, and $\{\pi_c^{(1)}, \pi_s^{(2)}, \pi_g^{(3)}\}$; and topic indicator variables $\mathbf{x}$ (parameters $\phi$ can be integrated out). The sampler alternatives between: (a) sampling cluster indices $x_{in}$ using Gibbs updates in the Chinese restaurant franchise (CRF) representation of the HDP; (b) sampling the weights at all three levels conditioned on $\mathbf{x}$ using the usual posterior of a DP[2].

**Sampling category assignments $\mathbf{z}$:** Given current instantiation of the stick-breaking weights, using a defining property of a DP, for each input $n$, we have:

$$(\theta_{1,n}, ..., \theta_{T,n}, \theta_{new,n}) \sim \text{Dir}(\alpha^{(1)}\pi_{\mathbf{z}_n,1}^{(1)}, ..., \alpha^{(1)}\pi_{\mathbf{z}_n,T}^{(1)}, \alpha^{(1)}\pi_{\mathbf{z}_n,new}^{(1)}) \tag{9}$$

Combining the above likelihood term with the CRP prior (Eq. 8), the posterior over the category assignment can be calculated as follows:

$$p(\mathbf{z}_n|\theta_n, \mathbf{z}_{-n}, \pi^{(1)}) \propto p(\theta_n|\pi^{(1)}, \mathbf{z}_n)p(\mathbf{z}_n|\mathbf{z}_{-n}), \tag{10}$$

where $\mathbf{z}_{-n}$ denotes variables $\mathbf{z}$ for all observations other than $n$. When computing the probability of placing $\theta_n$ under a newly created category, its parameters are sampled from the prior.

**Sampling DBM's hidden units:** Given the states of the DBM's top-level multinomial unit $\mathbf{h}^3$, conditional samples from $P(\mathbf{h}_n^1, \mathbf{h}_n^2|\mathbf{h}_n^3, \mathbf{v}_n)$ can be obtained by running a Gibbs sampler that alternates between sampling the states of $\mathbf{h}_n^1$ independently given $\mathbf{h}_n^2$, and vice versa. Conditioned on topic assignments $x_{in}$ and $\mathbf{h}_n^2$, the states of the multinomial unit $\mathbf{h}_n^3$ for each input $n$ are sampled using Gibbs conditionals:

$$P(\mathbf{h}_{in}^3|\mathbf{h}_n^2, \mathbf{h}_{-in}^3, \mathbf{x}_n) \propto P(\mathbf{h}_n^2|\mathbf{h}_n^3)P(\mathbf{h}_{in}^3|\mathbf{x}_{in}), \tag{11}$$

where the first term is given by the product of logistic functions (see Eq. 4):

$$P(\mathbf{h}^2|\mathbf{h}^3) = \prod_l P(h_l^2|\mathbf{h}^3), \quad \text{with} \quad P(h_l^2 = 1|\mathbf{h}^3) = \frac{1}{1 + \exp\left(-\sum_m \mathbf{W}_{lm}^{(3)}h_m^3\right)}, \tag{12}$$

and the second term $P(\mathbf{h}_{in}^3)$ is given by the multinomial: $\text{Mult}(\phi_{x_{in}})$ (see Eq. 7, in our conjugate setting, parameters $\phi$ can be further integrated out).

**Fine-tuning DBM:** More importantly, conditioned on $\mathbf{h}^3$, we can further *fine-tune* low-level DBM parameters $\psi = \{\mathbf{W}^{(1)}, \mathbf{W}^{(2)}, \mathbf{W}^{(3)}\}$ by applying approximate maximum likelihood learning (see section 2) to the conditional DBM model of Eq. 4. For the stochastic approximation algorithm, as the partition function depends on the states of $\mathbf{h}^3$, we maintain one "persistent" Markov chain per data point (for details see [22, 14]).

**Making predictions:** Given a test input $\mathbf{v}_t$, we can quickly infer the approximate posterior over $\mathbf{h}_t^3$ using the mean-field of Eq. 2, followed by running the full Gibbs sampler to get approximate samples from the posterior over the category assignments. In practice, for faster inference, we fix learned topics $\phi_t$ and approximate the marginal likelihood that $\mathbf{h}_t^3$ belongs to category $\mathbf{z}_t$ by assuming that document specific DP can be well approximated by the class-specific DP[3] $G_t \approx G_{\mathbf{z}_t}^{(1)}$ (see Fig. 1):

$$P(\mathbf{h}_t^3|\mathbf{z}_t, G^{(1)}, \phi) = \int_{G_t} P(\mathbf{h}_t^3|\phi, G_t)P(G_t|G_{\mathbf{z}_t}^{(1)})dG_t \approx P(\mathbf{h}_t^3|\phi, G_{\mathbf{z}_t}^{(1)}), \tag{13}$$

Combining this likelihood term with nCRP prior $P(\mathbf{z}_t|\mathbf{z}_{-t})$ (Eq. 8) allows us to efficiently infer approximate posterior over category assignments[4].

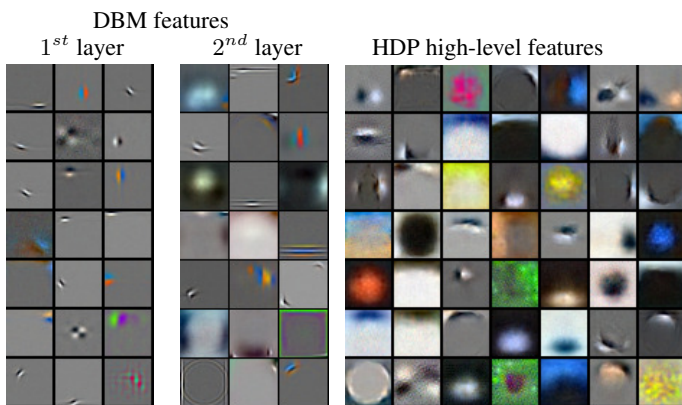

DBM features
$1^{st}$ layer    $2^{nd}$ layer    HDP high-level features

1. bed, chair, clock, couch, dinosaur, lawn mower, table, telephone, television, wardrobe
2. bus, house, pickup truck, streetcar, tank, tractor, train
3. crocodile, kangaroo, lizard, snake, spider, squirrel
4. hamster, mouse, rabbit, raccoon, possum, bear
5. apple, orange, pear, sunflower, sweet pepper
6. baby, boy, girl, man, woman
7. dolphin, ray, shark, turtle, whale
8. otter, porcupine, shrew, skunk
9. beaver, camel, cattle, chimpanzee, elephant
10. fox, leopard, lion, tiger, wolf
11. maple tree, oak tree, pine tree, willow tree
12 flatfish, seal, trout, worm
13 butterfly, caterpillar, snail
14 bee, crab, lobster
15 bridge, castle, road, skyscraper
16 bicycle, keyboard, motorcycle, orchid, palm tree
17 bottle, bowl, can, cup, lamp
18 cloud, plate, rocket  19. mountain, plain, sea
20 poppy, rose, tulip    21. aquarium fish, mushroom
22 beetle, cockroach    23. forest

Figure 2: A random subset of the $1^{st}$, $2^{nd}$ layer DBM features, and higher-level class-sensitive HDP features/topics.

Figure 3: A typical partition of the 100 basic-level categories

## 5   Experiments

We present experimental results on the CIFAR-100 [8], handwritten character [9], and human motion capture recognition datasets. For all datasets, we first pretrain a DBM model in unsupervised fashion on raw sensory input (e.g. pixels, or 3D joint angles), followed by fitting an HDP prior, which is run for 200 Gibbs sweeps. We further run 200 additional Gibbs steps in order to fine-tune parameters of the entire compound HDP-DBM model. This was sufficient to reach convergence and obtain good performance. Across all datasets, we also assume that the basic-level category labels are given, but no super-category labels are available. The training set includes many examples of familiar categories but only a few examples of a novel class. Our goal is to generalize well on a novel class.

In all experiments we compare performance of HDP-DBM to the following alternative models: stand-alone Deep Boltzmann Machines, Deep Belief Networks [5], "Flat HDP-DBM" model, that always uses a single super-category, SVMs, and k-NN. The Flat HDP-DBM approach could potentially identify a set of useful high-level features common to all categories. Finally, to evaluate performance of DBMs (and DBNs), we follow [14]. Note that using HDPs on top of raw sensory input (i.e. pixels, or even image-specific GIST features) performs far worse compared to HDP-DBM.

### 5.1   CIFAR-100 dataset

The CIFAR-100 image dataset [8] contains 50,000 training and 10,000 test images of 100 object categories (100 per class), with $32 \times 32 \times 3$ RGB pixels. Extreme variability in scale, viewpoint, illumination, and cluttered background makes object recognition task for this dataset quite difficult. Similar to [8], in order to learn good generic low-level features, we first train a two-layer DBM in completely unsupervised fashion using 4 million tiny images[5] [23]. We use a conditional Gaussian distribution to model observed pixel values [8, 6]. The first DBM layer contained 10,000 binary hidden units, and the second layer contained M=1000 softmax units, each defining a distribution over $10,000$ second layer features[6]. We then fit an HDP prior over $\mathbf{h}^2$ to the 100 object classes.

Fig. 2 displays a random subset of the $1^{st}$ and $2^{nd}$ layer DBM features, as well as higher-level class-sensitive features, or topics, learned by the HDP model. To visualize a particular higher-level feature, we first sample $M$ words from a fixed topic $\phi_t$, followed by sampling RGB pixel values from the conditional DBM model. While DBM features capture mostly low-level structure, including edges and corners, the HDP features tend to capture higher-level structure, including contours, shapes, color components, and surface boundaries. More importantly, features at all levels of the hierarchy evolve without incorporating any image-specific priors. Fig. 3 shows a typical partition over 100 classes that our model learns with many super-categories containing semantically similar classes.

We next illustrate the ability of the HDP-DBM to generalize from a single training example of a "pear" class. We trained the model on 99 classes containing 500 training images each, but only one training example of a "pear" class. Fig. 4 shows the kind of transfer our model is performing: the model discovers that pears are like apples and oranges, and not like other classes of images, such as dolphins, that reside in very different parts of the hierarchy. Hence the novel category can inherit

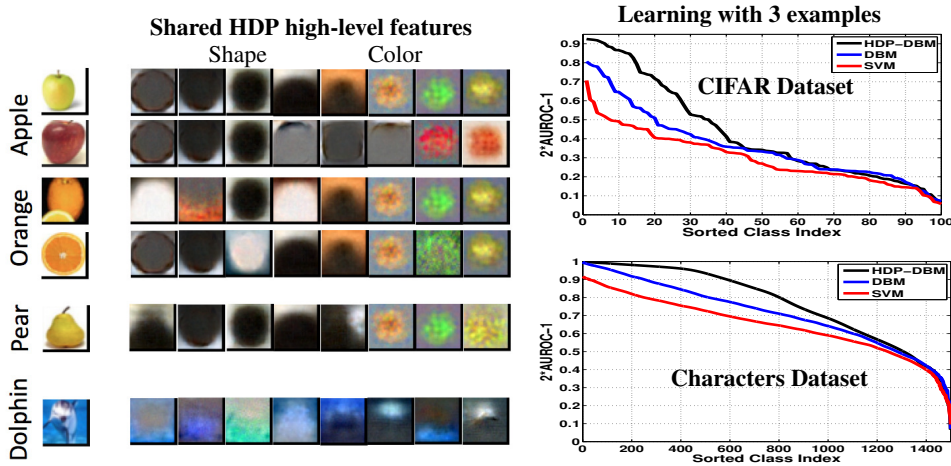

Figure 4: **Left:** Training examples along with eight most probable topics $\phi_t$, ordered by hand. **Right:** Performance of HDP-DBM, DBM, and SVMs for all object classes when learning with 3 examples. Object categories are sorted by their performance.

| | CIFAR Dataset Number of examples | | | | | Handwritten Characters Number of examples | | | | Motion Capture Number of examples | | | | |
|---|---|---|---|---|---|---|---|---|---|---|---|---|---|---|
| **Model** | 1 | 3 | 5 | 10 | 50 | 1 | 3 | 5 | 10 | 1 | 3 | 5 | 10 | 50 |
| Tuned HDP-DBM | **0.36** | **0.41** | **0.46** | **0.53** | **0.62** | **0.67** | **0.78** | **0.87** | **0.93** | **0.67** | **0.84** | **0.90** | **0.93** | **0.96** |
| HDP-DBM | 0.34 | 0.39 | **0.45** | **0.52** | **0.61** | 0.65 | 0.76 | 0.85 | **0.92** | **0.66** | 0.82 | 0.88 | **0.93** | **0.96** |
| Flat HDP-DBM | 0.27 | 0.37 | 0.42 | 0.50 | **0.61** | 0.58 | 0.73 | 0.82 | 0.89 | 0.63 | 0.79 | 0.86 | 0.91 | **0.96** |
| DBM | 0.26 | 0.36 | 0.41 | 0.48 | **0.61** | 0.57 | 0.72 | 0.81 | 0.89 | 0.61 | 0.79 | 0.85 | 0.91 | **0.95** |
| DBN | 0.25 | 0.33 | 0.37 | 0.45 | 0.60 | 0.51 | 0.72 | 0.81 | 0.89 | 0.61 | 0.79 | 0.84 | 0.92 | **0.96** |
| SVM | 0.18 | 0.27 | 0.31 | 0.38 | **0.61** | 0.41 | 0.66 | 0.77 | 0.86 | 0.54 | 0.78 | 0.84 | 0.91 | **0.96** |
| 1-NN | 0.17 | 0.18 | 0.19 | 0.20 | 0.32 | 0.43 | 0.65 | 0.73 | 0.81 | 0.58 | 0.75 | 0.81 | 0.88 | 0.93 |
| GIST | 0.27 | 0.31 | 0.33 | 0.39 | 0.58 | - | - | - | - | - | - | - | - |

Table 1: Classification performance on the test set using 2*AUROC-1. The results in bold correspond to ROCs that are statistically indistinguishable from the best (the difference is not statistically significant).

the prior distribution over similar high-level shape and color features, allowing the HDP-DBM to generalize considerably better to new instances of the "pear" class.

Table 1 quantifies performance using the area under the ROC curve (AUROC) for classifying 10,000 test images as belonging to the novel vs. all other 99 classes (we report 2*AUROC-1, so zero corresponds to the classifier that makes random predictions). The results are averaged over 100 classes using "leave-one-out" test format. Based on a single example, the HDP-DBM model achieves an AUROC of 0.36, significantly outperforming DBMs, DBNs, SVMs, as well as 1-NN using standard image-specific GIST features [24] that achieve an AUROC of 0.26, 0.25, 0.18 and 0.27 respectively. Table 1 also shows that fine-tuning parameters of *all layers jointly* as well as learning super-category hierarchy significantly improves model performance. As the number of training examples increases, the HDP-DBM model still consistently outperforms alternative methods. Fig. 4 further displays performance of HDP-DBM, DBM, and SVM models for all object categories when learning with only three examples. Observe that over 40 classes benefit in various degrees from learning a hierarchy.

## 5.2 Handwritten Characters

The handwritten characters dataset [9] can be viewed as the "transpose" of MNIST. Instead of containing 60,000 images of 10 digit classes, the dataset contains 30,000 images of 1500 characters (20 examples each) with $28 \times 28$ pixels. These characters are from 50 alphabets from around the world, including Bengali, Cyrillic, Arabic, Sanskrit, Tagalog (see Fig. 5). We split the dataset into 15,000 training and 15,000 test images (10 examples of each class). Similar to the CIFAR dataset, we pre-train a two-layer DBM model, with the first layer containing 1000 hidden units, and the second layer containing M=100 softmax units, each defining a distribution over 1000 second layer features.

Fig. 2 displays a random subset of training images, along with the $1^{st}$ and $2^{nd}$ layer DBM features, as well as higher-level class-sensitive HDP features. The HDP features tend to capture higher-level parts, many of which resemble pen "strokes". Table 1 further shows results for classifying 15,000 test images as belonging to the novel vs. all other 1,499 character classes. The HDP-DBM model significantly outperforms other methods, particularly when learning characters with few training examples. Fig. 6 further displays learned super-classes along with examples of *entirely novel* characters that have been generated by the model for the same super-class, as well as conditional samples

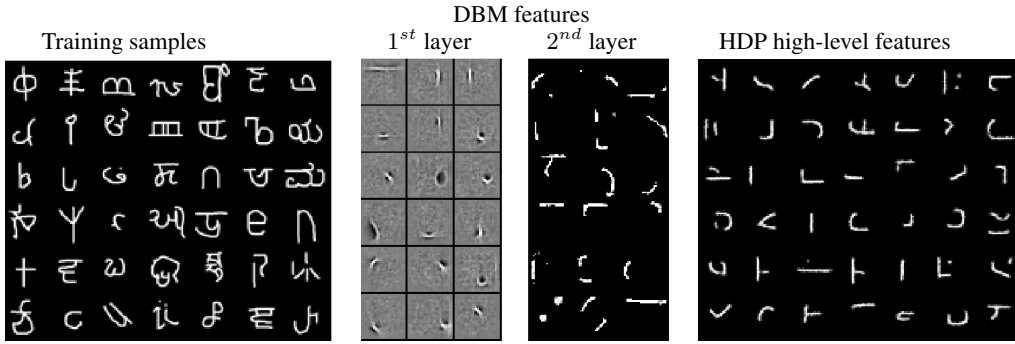

Figure 5: A random subset of the training images along with $1^{st}$ and $2^{nd}$ layer DBM features, as well as higher-level class-sensitive HDP features/topics.

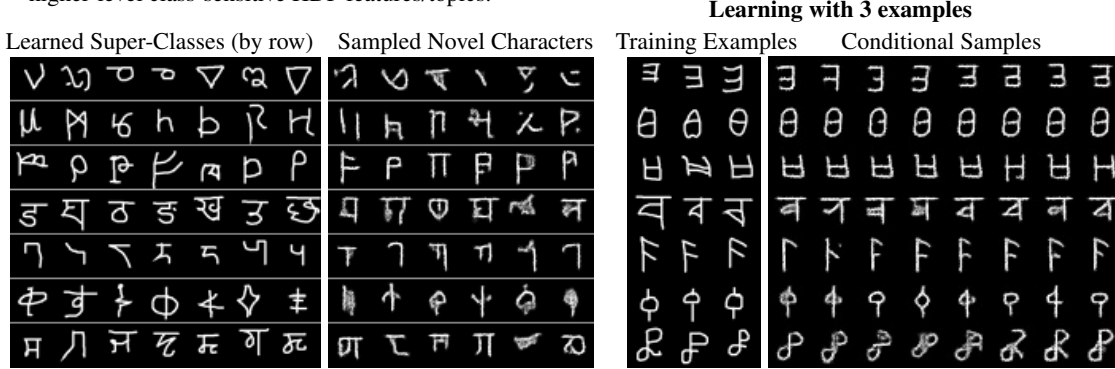

Figure 6: **Left:** Learned super-classes along with examples of novel characters, generated by the model for the same super-class. **Right:** Three training examples along with 8 conditional samples.

when learning only with three training examples. (we note that using Deep Belief Networks instead of DBMs produced far inferior generative samples). Remarkably, many samples look realistic, containing coherent, long-range structure, while at the same time being different from existing training images (see Supplementary Materials for a much richer class of generated samples).

### 5.3 Motion capture

We next applied our model to human motion capture data consisting of sequences of 3D joint angles plus body orientation and translation [18]. The dataset contains 10 walking styles, including normal, drunk, graceful, gangly, sexy, dinosaur, chicken, old person, cat, and strong. There are 2500 frames of each style at 60fps, where each time step was represented by a vector of 58 real-valued numbers. The dataset was split at random into 1500 training and 1000 test frames of each style. We further preprocessed the data by treating each window of 10 consecutive frames as a single $58 * 10 = 580$-d data vector. For the two-layer DBM model, the first layer contained 500 hidden units, with the second layer containing $M=50$ softmax units, each defining a distribution over 500 second layer features. As expected, Table 1 shows that the HDP-DBM model performs much better compared to other models when discriminating between existing nine walking styles vs. novel walking style. The difference is particularly large in the regime when we observe only a handful number of training examples of a novel walking style.

## 6 Conclusions

We developed a compositional architecture that learns an HDP prior over the activities of top-level features of the DBM model. The resulting compound HDP-DBM model is able to learn low-level features from raw sensory input, high-level features, as well as a category hierarchy for parameter sharing. Our experimental results show that the proposed model can acquire new concepts from very few examples in a diverse set of application domains. The compositional model considered in this paper was directly inspired by the architecture of the DBM and HDP, but it need not be. Indeed, any other deep learning module, including Deep Belief Networks, sparse auto-encoders, or any other hierarchical Bayesian model can be adapted. This perspective opens a space of compositional models that may be more suitable for capturing the human-like ability to learn from few examples.

**Acknowledgments**: This research was supported by NSERC, ONR (MURI Grant 1015GNA126), ONR N00014-07-1-0937, ARO W911NF-08-1-0242, and Qualcomm.

## Footnotes

[1]For clarity, we use three hidden layers. Extensions to models with more than three layers is trivial.

[2]Conditioned on the draw of the super-class DP $G_s^{(2)}$ and the state of the CRF, the posteriors over $G_c^{(1)}$ become independent. We can easily speed up inference by sampling from these conditionals in parallel.

[3]We note that $G_{\mathbf{z}_t}^{(1)} = \text{E}[G_t|G_{\mathbf{z}_t}^{(1)}]$

[4]In all of our experimental results, computing this approximate posterior takes a fraction of a second.

[5]The dataset contains random images of natural scenes downloaded from the web

[6]We also experimented with a 3-layer DBM model, as well as various softmax parameters: $M = 500$ and $M = 2000$. The difference in performance was not significant.

## References

[1] E. Bart, I. Porteous, P. Perona, and M. Welling. Unsupervised learning of visual taxonomies. In *CVPR*, pages 1–8, 2008.

[2] David M. Blei, Thomas L. Griffiths, and Michael I. Jordan. The nested chinese restaurant process and bayesian nonparametric inference of topic hierarchies. *J. ACM*, 57(2), 2010.

[3] Kevin R. Canini and Thomas L. Griffiths. Modeling human transfer learning with the hierarchical dirichlet process. In *NIPS 2009 workshop: Nonparametric Bayes*, 2009.

[4] Li Fei-Fei, R. Fergus, and P. Perona. One-shot learning of object categories. *IEEE Trans. Pattern Analysis and Machine Intelligence*, 28(4):594–611, April 2006.

[5] G. E. Hinton, S. Osindero, and Y. W. Teh. A fast learning algorithm for deep belief nets. *Neural Computation*, 18(7):1527–1554, 2006.

[6] G. E. Hinton and R. R. Salakhutdinov. Reducing the dimensionality of data with neural networks. *Science*, 313(5786):504 – 507, 2006.

[7] C. Kemp, A. Perfors, and J. Tenenbaum. Learning overhypotheses with hierarchical Bayesian models. *Developmental Science*, 10(3):307–321, 2006.

[8] Alex Krizhevsky. Learning multiple layers of features from tiny images. Technical report, Dept. of Computer Science, University of Toronto, 2009.

[9] Brenden Lake, Ruslan Salakhutdinov, Jason Gross, and Josh Tenenbaum. One-shot learning of simple visual concepts. In *Proceedings of the 33rd Annual Conference of the Cognitive Science Society*, 2011.

[10] H. Larochelle, Y. Bengio, J. Louradour, and P. Lamblin. Exploring strategies for training deep neural networks. *Journal of Machine Learning Research*, 10:1–40, 2009.

[11] Honglak Lee, Roger Grosse, Rajesh Ranganath, and Andrew Y. Ng. Convolutional deep belief networks for scalable unsupervised learning of hierarchical representations. In *Proceedings of the 26th International Conference on Machine Learning*, pages 609–616, 2009.

[12] M. A. Ranzato, Y. Boureau, and Y. LeCun. Sparse feature learning for deep belief networks. *Advances in Neural Information Processing Systems*, 2008.

[13] A. Rodriguez, D. Dunson, and A. Gelfand. The nested Dirichlet process. *Journal of the American Statistical Association*, 103:11311144, 2008.

[14] R. R. Salakhutdinov and G. E. Hinton. Deep Boltzmann machines. In *Proceedings of the International Conference on Artificial Intelligence and Statistics*, volume 12, 2009.

[15] R. R. Salakhutdinov and G. E. Hinton. Replicated softmax: an undirected topic model. In *Advances in Neural Information Processing Systems*, volume 22, 2010.

[16] L.B. Smith, S.S. Jones, B. Landau, L. Gershkoff-Stowe, and L. Samuelson. Object name learning provides on-the-job training for attention. *Psychological Science*, pages 13–19, 2002.

[17] E. B. Sudderth, A. Torralba, W. T. Freeman, and A. S. Willsky. Describing visual scenes using transformed objects and parts. *International Journal of Computer Vision*, 77(1-3):291–330, 2008.

[18] G. Taylor, G. E. Hinton, and S. T. Roweis. Modeling human motion using binary latent variables. In *Advances in Neural Information Processing Systems*. MIT Press, 2006.

[19] Y. W. Teh and G. E. Hinton. Rate-coded restricted Boltzmann machines for face recognition. In *Advances in Neural Information Processing Systems*, volume 13, 2001.

[20] Y. W. Teh and M. I. Jordan. Hierarchical Bayesian nonparametric models with applications. In *Bayesian Nonparametrics: Principles and Practice*. Cambridge University Press, 2010.

[21] Y. W. Teh, M. I. Jordan, M. J. Beal, and D. M. Blei. Hierarchical dirichlet processes. *Journal of the American Statistical Association*, 101(476):1566–1581, 2006.

[22] T. Tieleman. Training restricted Boltzmann machines using approximations to the likelihood gradient. In *ICML*. ACM, 2008.

[23] A. Torralba, R. Fergus, and W. T. Freeman. 80 million tiny images: a large dataset for nonparametric object and scene recognition. *IEEE Transactions on Pattern Analysis and Machine Intelligence*, 30(11):1958–1970, 2008.

[24] A. Torralba, R. Fergus, and Y. Weiss. Small codes and large image databases for recognition. In *Proceedings of the IEEE Conference on Computer Vision and Pattern Recognition*, 2008.

[25] Fei Xu and Joshua B. Tenenbaum. Word learning as bayesian inference. *Psychological Review*, 114(2), 2007.

[26] L. Younes. On the convergence of Markovian stochastic algorithms with rapidly decreasing ergodicity rates, March 17 2000.

